# An Application of Tree-Structured Expectation Propagation for Channel Decoding

**Pablo M. Olmos**[*], **Luis Salamanca**[*], **Juan J. Murillo-Fuentes**[*], **Fernando Pérez-Cruz**[†]
[*] Dept. of Signal Theory and Communications, University of Sevilla
41092 Sevilla Spain
{olmos,salamanca,murillo}@us.es
[†] Dept. of Signal Theory and Communications, University Carlos III in Madrid
28911 Leganés (Madrid) Spain
fernando@tsc.uc3m.es

## Abstract

We show an application of a tree structure for approximate inference in graphical models using the expectation propagation algorithm. These approximations are typically used over graphs with short-range cycles. We demonstrate that these approximations also help in sparse graphs with long-range loops, as the ones used in coding theory to approach channel capacity. For asymptotically large sparse graph, the expectation propagation algorithm together with the tree structure yields a completely disconnected approximation to the graphical model but, for for finite-length practical sparse graphs, the tree structure approximation to the code graph provides accurate estimates for the marginal of each variable. Furthermore, we propose a new method for constructing the tree structure on the fly that might be more amenable for sparse graphs with general factors.

## 1 Introduction

Belief propagation (BP) has become the standard procedure to decode channel codes, since in 1996 MacKay [7] proposed BP to decode codes based on low-density parity-check (LDPC) matrices with linear complexity. A rate $r = k/n$ LDPC code can be represented as a sparse factor graph with $n$ variable nodes (typically depicted on the left side) and $n - k$ factor nodes (on the right side), in which the number of edges is linear in $n$ [15]. The first LDPC codes [6] presented a regular structure, in which all variables and factors had, respectively, $\ell$ and $r$ connections, i.e. an $(\ell, r)$ LDPC code. But the analysis of their limiting decoding performance, when $n$ tends to infinity for a fixed rate, showed that they do not approach the channel capacity [15]. To improve the performance of regular LDPC codes, we can define an (irregular) LDPC ensemble as the set of codes randomly generated according to the degree distribution (DD) from the edge perspective as follows:

$$\lambda(x) = \sum_{i=1}^{l_{\max}} \lambda_i x^{i-1} \qquad \text{and} \qquad \rho(x) = \sum_{j=1}^{r_{\max}} \rho_j x^{j-1},$$

where the fraction of edges with left degree $i$ (from variables to factors) is given by $\lambda_i$ and the fraction of edges with right degree $j$ (from factors to variables) is given by $\rho_j$. The left (right) degree of an edge is the degree of the variable (factor) node it is connected to. The rate of the code is then given by $r = 1 - \int_0^1 \rho(x)dx / \int_0^1 \lambda(x)dx$, and the total number of edges by $E = n/(\sum_i \lambda_i/i)$.

Although optimized irregular LDPC codes can achieve the channel capacity with a decoder based on BP [15], they present several drawbacks. First, the error floor in those codes increases significantly, because capacity achieving LDPC ensembles with BP decoding have a large fraction of variables

with two connections and they present low minimum distances. Second, the maximum number of ones per column $l_{\max}$ tends to infinity to approach capacity. These problems limit the BP decoding performance of capacity approaching codes, when we work with finite length codes used in real applications.

Approximate inference in graphical models can be solved using more accurate methods that significantly improve the BP performance, especially for dense graphs with short-range loops. A non-exhaustive list of methods are: generalized BP [22], expectation propagation (EP) [10], fractional BP [19], linear programming [17] and power EP [8]. A detailed list of contributions for approximate inference can be found in [18] and the references therein. But it is a common belief that BP is sufficiently accurate to decode LDPC codes and other approximate inference algorithms would not outperform BP decoding significantly, if at all. In this paper, we challenge that belief and show that more accurate approximate inference algorithms for graphical models can also improve the BP decoding performance for LDPC codes, which are sparse graphical models with long-range loops.

We particularly focus on tree-structured approximations for inference in graphical models [9] using the expectation propagation (EP) algorithm, because it presents a simple algorithmic implementation for LDPC decoding transmitted over the binary erasure channel (BEC)[1], although other higher order inference algorithms might be suitable for this problem, as well, as in [20] it was proven a connection between some of them. We show the results for the BEC, because it has a simple structure amenable for deeper analysis and most of its properties carry over to actual communications channels [14].

The EP with a tree-structured approximation can be presented in a similar way as the BP decoder for an LDPC code over the BEC [11], with similar run-time complexity. We show that a decoder based on EP with a tree-structured approximation converges to the BP solution for the asymptotic limit n → ∞, for finite-length graphs the performance is otherwise improved significantly [13, 11]. For finite graphs, the presence of cycles in the graph degrades the BP estimate and we show that the EP solution with a tree-structured approximation is less sensitive to the presence of such loops, and provides more accurate estimates for the marginal of each bit. Therefore, it makes the expectation propagation with a tree-structured approximation (for short we refer to this algorithm as tree-structured EP or TEP) a more practical decoding algorithm for finite length LDPC codes.

Besides, the analysis of the application of the tree-structured EP to channel decoding over the BEC leads to another way of fixing the approximating tree structure different from the one proposed in [9] for dense codes with positive correlation potentials. In channel coding, the factors of the graph are parity-checks and the correlations are high but can change from positive to negative by the flip of a single variable. Therefore, the pair-wise mutual information is zero for any two variables (unless the factor only contains two variables) and we could not define a prefixed tree structure with the algorithm in [9]. In contrast, we propose a tree structure that is learnt on the fly based on the graph itself, hence it might be amenable for other potentials and sparser graphs.

The rest of the paper is organized as follows. In Section 2, we present the peeling decoder, which is the interpretation of the BP algorithm for LDPC codes over the BEC, and how it can be extended to incorporate the tree-structured EP decoding procedure. In Section 3, we analyze the TEP decoder performance for LDPC codes in both the asymptotic and the finite-length regimen. We provide an estimation of the TEP decoder error rate for a given LDPC ensemble. We conclude the paper in Section 5.

## 2 Tree-structured EP and the peeling decoder

The BP algorithm was proposed as a message passing algorithm [5] but, for the BEC, it exhibits a simpler formulation, in which the non-erased variable nodes are removed from the graph in each iteration [4], because we either have absolute certainty about the received bit (0 or 1) or complete ignorance (?). The BP under this interpretation is referred to as the peeling decoder (PD) [3, 15] and it is easily described using the factor graph of the code. The first step is to initialize the graph by removing all the variable nodes corresponding to non-erased bits. When removing a one-valued non-erased variable node, the parity of the factors it was connected to are flipped. After the initialization

stage, the algorithm proceeds over the resulting graph by removing a factor and a variable node in each step:

1. It looks for any factor linked to a single variable (a check node of degree one). The peeling decoder copies the parity of this factor into the variable node and removes the factor.

2. It removes the variable node that we have just de-erased. If the variable was assigned a one, it changes the parity of the factors it was connected to.

3. It repeats Steps 1 and 2 until all the variable nodes have been removed, successful decoding, or until there are no degree-one factors left, unsuccessful decoding.

We illustrate an example of the PD for a 1/2-rate code with four variables in Figure 1. The first and last bits have not been erased and when we remove them from the graph, the second factor is singled connected to the third variable, which can be now de-erased (Figure 1(b)). Finally, the first factor is singled connected to the second variable, decoding the transmitted codeword (Figure 1(c)).

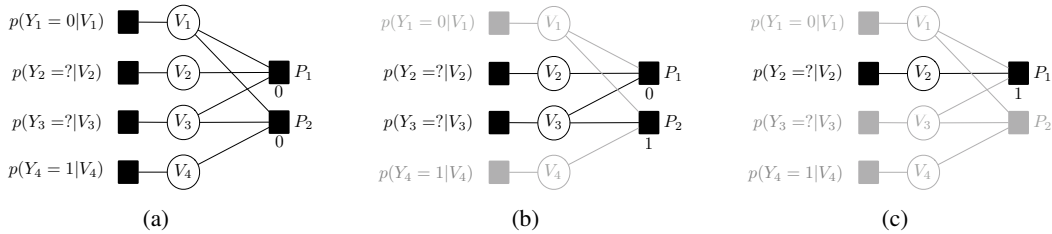

Figure 1: Example of the PD algorithm for LDPC channel decoding in the erasure channel.

The analysis of the PD for fixed-rate codes, proposed in [3, 4], allows to compute its threshold in the BEC. This result can be used to optimize the DD to build irregular LDPC codes that, as n tends to infinity, approach the channel capacity. However, as already discussed, these codes present higher error floors, because they present many variables with only two edges, and they usually present poor finite-length performance due to the slow convergence to the asymptotic limit [15].

## 2.1 The TEP decoder

The tree-structured EP overlaps a tree over the variables on the graph to further impose pairwise marginal constraints. In the procedure proposed in [9] the tree was defined by measuring the mutual information between a pair of variables, before running the EP algorithm. The mutual information between pair of variables is zero for parity-check factors with more than two variables and we need to define the structure in another way. We propose to define the tree structure on the fly. Let's assume that we run the PD in the previous section and yields an unsuccessful decoding. Any factor of degree two in the remaining graph either tells us that the connected variables are equal (if the parity check is zero), or opposite (if the parity check is one). We should link these two variables by the tree structure, because their marginal would provide further information to the remaining erased variables in the graph. The proposed algorithm actually replaces one variable by the other and iterates until a factor of degree one is created and more variables can be de-erased. When this happens a tree structure has been created, in which the pairwise marginal constraint provides information that was not available with single marginals approximations.

The TEP decoder can be explained in a similar fashion as the PD decoder, in which instead of looking for degree-one factors, we look for degree one and two. We initialize the TEP decoder, as the PD, by removing all known variable nodes and updating the parity checks for the variables that are one. The TEP then removes a variable and a factor per iteration:

1. It looks for a factor of degree one or two.

2. If a factor of degree one is found, the TEP recovers the associated variable, performing the Steps 1 and 2 of the PD previously described.

3. If a factor of degree two is found, the decoder removes it from the graph together with one of the variable nodes connected to it and the two associated edges. Then, it reconnects

to the remaining variable node all the factors that were connected to the removed variable node. The parities of the factors re-connected to the remaining variable node are reversed if the removed factor had parity one.

4. Steps 1-3 are repeated until all the variable nodes have been removed, successful decoding, or the graph runs out of factors of degree one or two, unsuccessful decoding.

The process of removing a factor of degree two is sketched in Figure 2. First, the variable $V_1$ heirs the connections from $V_2$ (solid lines), see Figure 2(b). Finally, the factor $P_1$ and the variable $V_2$ can be removed (Figure 2(c)), because they have no further implication in the decoding process. $V_2$ is de-erased once $V_1$ is de-erased. The TEP removes a factor and a variable node per iteration, as the PD does. The removal of a factor and a variable does not increase the complexity of the TEP decoder compared to the BP algorithm. Both TEP and BP algorithms have complexity $\mathcal{O}(\mathtt{n})$.

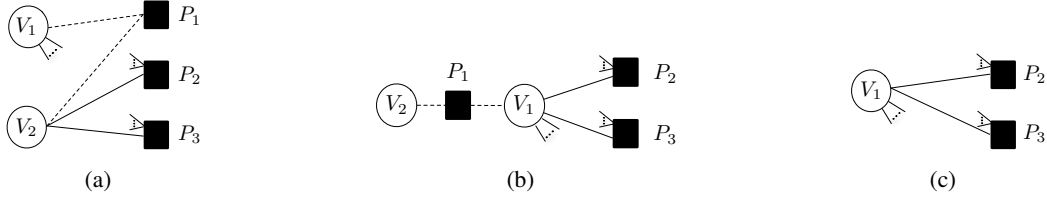

(a)     (b)     (c)

Figure 2: In (a) we show two variable nodes, $V_1$ and $V_2$, that share a factor of degree two $P_1$. In (b), $V_1$ heirs the connections of $V_2$ (solid lines). In (c), we show the graph once $P_1$ and $V_2$ have been removed. If $P_1$ is parity one, the parities of $P_2$, $P_3$ are reversed.

By removing factors of degree two, we eventually create factors of degree one, whenever we find an scenario equivalent to the one depicted in Figure 3. Consider two variable nodes connected to a factor of degree two that also share another factor with degree three, as illustrated in Figure 3(a). When we remove the factor $P_3$ and the variable node $V_2$, the factor $P_4$ is now degree one, as illustrated in Figure 3(b). At the beginning of the decoding algorithm, it is unlikely that the two variable nodes in a factor of degree two also share a factor of degree three. However, as we remove variables and factors, the probability of this event grows.

Note that, when we remove a factor of degree two connected to variables $V_1$ and $V_2$, in terms of the EP algorithm, we are including a pairwise factor between both variables. Therefore, the TEP equivalent tree structure is not fixed a priori and we construct it along the decoding process. Also, the steps of the TEP decoder can be presented as a linear combination of the columns of the parity-check matrix of the code and hence its solution is independent of the processing order.

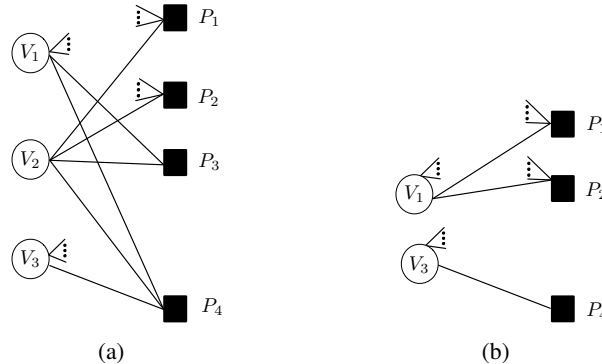

(a)     (b)

Figure 3: In (a), the variables $V_1$ and $V_2$ are connected to a degree two factor, $P_3$, and they also share a factor of degree three, $P_4$. In (b) we show the graph once the TEP has removed $P_3$ and $V_2$.

## 3   TEP analysis: expected graph evolution

We now sketch the proof of why the TEP decoder outperforms BP. The actual proof can be found in [12] (available as supplementary material). Both the PD and the TEP decoder sequentially reduces

the LDPC graph by removing check nodes of degree one or two. As a consequence, the decoding process yields a sequence of residual graphs and their associated DD. The DD sequence of the residual graphs constitutes a sufficient statistic to analyze this random process [1]. In [3, 4], the sequence of residual graphs follows a typical path or expected evolution [15]. The authors make use of Wormald's theorem in [21] to describe this path as the solution of a set of differential equations and characterized the typical deviation from it. For the PD, we have an analytical form for the evolution of the number of degree one factor as the decoding progresses, $r_1(\tau, \epsilon)$, as a function of the decoding time, $\tau$, and the erasure rate, $\epsilon$. The PD threshold $\epsilon_{\text{BP}}$ is the maximum $\epsilon$ for which $r_1(\tau, \epsilon) \geq 0, \forall \tau$. In [1, 15], the authors show that particular decoding realizations are Gaussian distributed around $r_1(\tau, \epsilon)$, with a variance of order $\alpha_{\text{BP}}/\text{n}$, where $\alpha_{\text{BP}}$ can be computed from the LDPC DD. They also provide the following approximation to the block error probability of elements of an LDPC ensemble:

$$\mathbb{E}_{\text{LDPC }[\lambda(x), \rho(x), \text{n}]} \left[ P_{\text{W}}^{\text{BP}}(\text{C}, \epsilon) \right] \approx \mathcal{Q} \left( \frac{\sqrt{\text{n}}(\epsilon_{\text{BP}} - \epsilon)}{\alpha_{\text{BP}}} \right), \tag{1}$$

where $P_{\text{W}}^{\text{BP}}(\text{C}, \epsilon)$ is the average block error probability for the code $\text{C} \in \text{LDPC } [\lambda(x), \rho(x), \text{n}]$. For the TEP decoder the analysis follows a similar path, but its derivation is more involved. For arbitrarily large codes, the expected graph evolution during the TEP decoding is computed in [12], with a set of non-linear differential equations. They track down the expected progression of the fraction of edges with left degree $i$, $l_i(\tau)$ for $i = 1, \ldots, l_{max}$, and right degree $j$, $r_j(\tau)$ for $j = 1, \ldots, r_{max}$ as the TEP decoder performs, where $\tau$ is a normalized time: if $u$ is the TEP iteration index and $E$ is the total number of edges in the original graph, then $\tau = u/E$. By Wormald's theorem [21], any real decoding realization does not differ from the solution of such equations in a factor larger than $\mathcal{O}(E^{-1/6})$. The TEP threshold, $\epsilon_{\text{TEP}}$, is found as the maximum erasure rate $\epsilon$ such that

$$r_{TEP}(\tau) \doteq r_1(\tau) + r_2(\tau) > 0, \qquad \forall \tau \in [0, n/E], \tag{2}$$

where $r_{TEP}(\tau)$ is computed by solving the system of differential equations in [12] and $\epsilon_{\text{TEP}} \geq \epsilon_{\text{BP}}$. Let us illustrate the accuracy of the model derived to analyze the TEP decoder properties. In Figure 4(a), for a regular $(3, 6)$ code with $\text{n} = 2^{17}$ and $\epsilon = 0.415$, we compare the solution of the system of differential equations for $R_1(\tau) = r_1(\tau)E$ and $R_2(\tau) = r_2(\tau)E$, depicted by thick solid lines, with 30 simulated decoding trajectories, depicted by thin dashed lines. We can see that empirical curves are tightly distributed around the predicted curves. Indeed, the distribution tends very quickly to $\text{n}$ to a Gaussian [1, 15]. All curves are plotted with respect the evolution of the normalized size of the graph at each time instant, denoted by $e(\tau)$ so that the decoding process starts on the right $e(\tau = 0) \approx 0.415$ and, if successful, finishes at $e(\tau_{\text{END}}) = 0$. In Figure 4(b) we reproduce, with identical conclusions, the same experiment for the irregular DD LDPC code defined by:

$$\lambda(x) = \frac{1}{6}x + \frac{5}{6}x^3, \tag{3}$$
$$\rho(x) = x^5. \tag{4}$$

For the TEP decoder to perform better than the BP decoder, it needs to significantly increase the number of check nodes of degree one that are created, which happens if two variables nodes share a degree-two check together along with a degree-three check node, as illustrated earlier in Figure 3(a). In [12], we compute the probability that two variable nodes that share a check node of degree-two also share another check node (scenario $\mathcal{S}$). If we randomly choose a particular degree-two check node at time $\tau$, the probability of scenario $\mathcal{S}$ is:

$$P_{\mathcal{S}}(\tau) = \frac{(l_{avg}(\tau) - 1)^2 (r_{avg}(\tau) - 1)}{e(\tau)E}, \tag{5}$$

where $l_{avg}(\tau)$ and $r_{avg}(\tau)$ are, respectively, the average left and right edge degrees, and $e(\tau)$ is the fraction of remaining edges in the graph. As the TEP decoder progresses, $l_{avg}(\tau)$ increases, because the remaining variables in the graph inherits the connections of the variables that have been removed, and $e(\tau)$ decreases, therefore creating new factors of degree one and improving the BP/PD performance. However, note that in the limit $\text{n} \to \infty$, $P_{\mathcal{S}}(\tau = 0) = 0$. Therefore, to improve the PD solution in this regime we require that $l_{avg}(\tau') \to \infty$ for some $\tau'$. The solution of the TEP decoder differential equations does not satisfy this property. For instance, in Figure 5 (a), we plot the expected evolution of $r_1(\tau)$ and $r_2(\tau)$ for $\text{n} \to \infty$ and the $(3, 6)$ regular LDPC ensemble when

we are just above the BP threshold for this code, which is $\epsilon_{\mathrm{BP}} \approx 0.4294$. Unlike Figure 4(a), $r_1(\tau)$ and $r_2(\tau)$ go to zero before $e(\tau)$ cancels: the TEP decoder gets stuck before completing the decoding process. In Fig.5 (b), we include the computed evolution for $l_{avg}(\tau)$. As shown, the fraction of degree two check nodes vanishes before $l_{avg}(\tau)$ becomes infinite. We conclude that, in the asymptotic limit $\mathrm{n} \to \infty$, the EP with tree-structure is not able to outperform the BP solution, which is optimal since LDPC codes become cycle free [15].

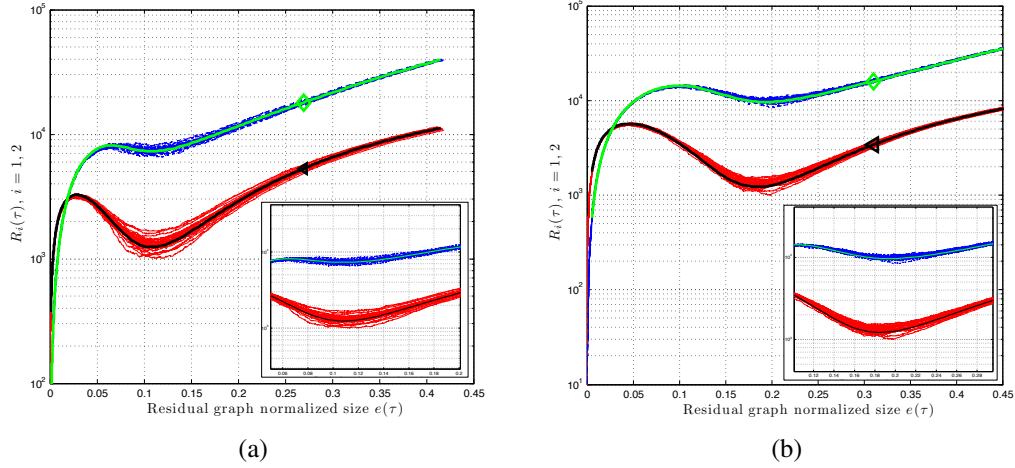

(a)          (b)

Figure 4: In (a), for a regular $(3,6)$ code with $\mathrm{n} = 2^{17}$ and $\epsilon = 0.415$, we compare the solution of the system of differential equations for $R_1(\tau) = r_1(\tau)E$ ($\triangleleft$) and $R_2(\tau) = r_2(\tau)E$ ($\diamond$) (thick solid lines) with 30 simulated decoding trajectories (thin dashed lines). In (b), we reproduce the same experiment for the irregular LDPC in (3) and (4) for $\epsilon = 0.47$.

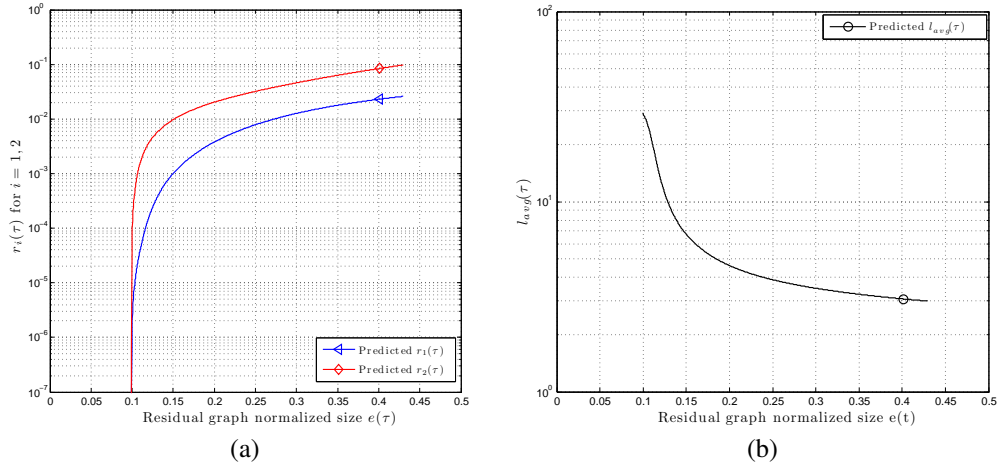

(a)          (b)

Figure 5: For the regular $(3,6)$ ensemble and $\epsilon_{\mathrm{BP}} \approx 0.4294$, in (a) we plot the expected evolution of $r_1(\tau)$ and $r_2(\tau)$ for $\mathrm{n} \to \infty$. In (b), we include the computed evolution of $l_{avg}(\tau)$ for this case.

## 3.1 Analysis in the finite-length regime

In the finite-length regime, the TEP decoder emerges as a powerful decoding algorithm. At a complexity similar to BP, i.e. of order $\mathcal{O}(\mathrm{n})$, it is able to further improve the BP solution thanks to a more accurate estimate of the marginal for each bit. We illustrate the TEP decoder performance for some regular and irregular finite-length LDPC codes. We first consider a rate $1/2$ regular $(3,6)$ LDPC code. This ensemble has no asymptotic error floor [15] and we plot the word error rate obtained with the TEP and the BP decoders with different code lengths in Figure 6(a). In Figure 6(b), we

include the results for the irregular DD in (3) and (4), where we can see that in all cases BP and TEP converge to the same error floor but, as in previous examples, the TEP decoder provides significant gains in the waterfall region and they are more significant for shorter codes.

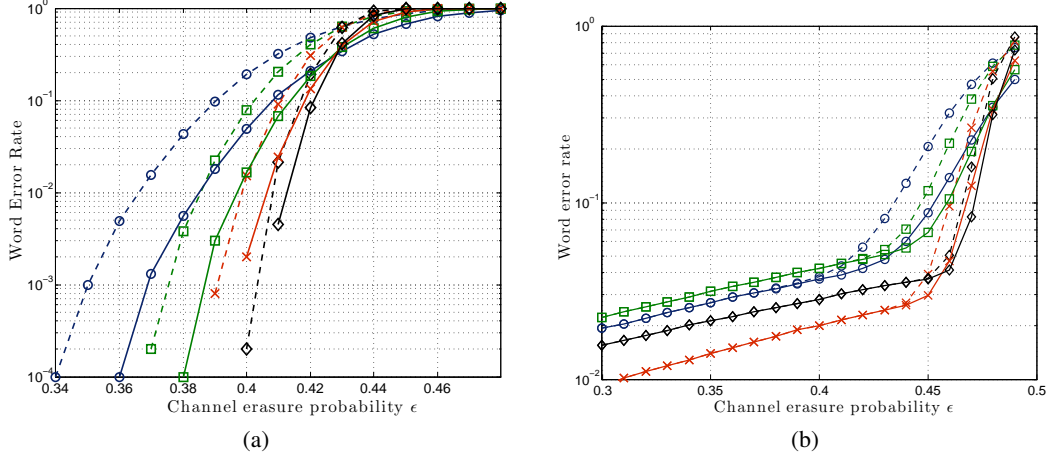

(a)   (b)

Figure 6: TEP (solid line) and BP (dashed line) decoding performance for a regular LDPC (3,6) code in (a), and the irregular LDPC in (3) and (4) in (b), with code lengths $n = 2^9$ ($\circ$), $n = 2^{10}$ ($\square$), $n = 2^{11}$ ($\times$) and $2^{12}$ ($\triangleright$).

The expected graph evolution during the TEP decoding in [12], which provides the average presence in the graph of degree one and two check nodes as the decoder proceeds, can be used to derive a coarse estimation of the TEP decoder probability of error for a given LDPC ensemble, similar to (1) for the BP decoder. By using the regular $(3,6)$ code as an example, in Figure 5(a), we plot the solution for $r_1(\tau)$ in the case $n \to \infty$. Let $\tau^*$ be the time at which the decoder gets stuck, i.e. $r_1(\tau^*) + r_2(\tau^*) = 0$. In Figure 7, we plot the solution for the evolution of $r_1(\tau, n, \epsilon_{\text{BP}})$ with respect to $e(t)$ for a $(3,6)$ regular code at $\epsilon = \epsilon_{\text{BP}} = \epsilon_{\text{TEP}}$. To avoid confusion, in the following we explicitly include the dependence with $n$ and $\epsilon$ in $r_1(\tau, n, \epsilon)$. The code lengths considered are $n = 2^{12}$ (+), $n = 2^{13}$ ($\circ$), $n = 2^{14}$ ($\square$), $n = 2^{15}$ ($\diamond$), $n = 2^{16}$ ($\times$) and $n = 2^{17}$ ($\bullet$). For finite-length values, we observe that $r_1(\tau^*, n, \epsilon_{\text{BP}})$ is not zero and, indeed, a closer look shows that the following approximation is reasonable tight:

$$r_1(\tau^*, n, \epsilon_{\text{TEP}}) \approx \gamma_{\text{TEP}} n^{-1}, \tag{6}$$

where we compute $\gamma_{\text{TEP}}$ from the ensemble. For the $(3,6)$ regular case, we obtain $\gamma_{\text{TEP}} \approx 0.3198$ [12]. The idea to estimate the TEP decoder performance at $\epsilon = \epsilon_{\text{BP}} + \Delta\epsilon$ is to assume that any particular realization will succeed almost surely as long as the fraction of degree one check nodes at $\tau^*$ is positive. For $\epsilon = \epsilon_{\text{BP}} + \Delta\epsilon$, we can approximate $r_1(\tau^*, n, \epsilon)$ as follows:

$$r_1(\tau^*, n, \epsilon) = \left. \frac{\partial r_1(\tau, n, \epsilon)}{\partial\epsilon} \right|_{\substack{\tau=\tau^* \\ \epsilon=\epsilon_{\text{TEP}}}} \Delta\epsilon + \gamma_{\text{TEP}} n^{-1}. \tag{7}$$

In [1, 15], it is shown that simulated trajectories for the evolution of degree one check nodes under BP are asymptotically Gaussian distributed and this is observed for the TEP decoder as well. Furthermore, the variance is of order $\delta(\tau)/n$, where $\delta(\tau)$ depends on the ensemble and the decoder [1]. To estimate the TEP decoder error rate, we compute the probability that the fraction of degree one check nodes at at $\tau^*$ is positive. Since it is distributed as $\mathcal{N}(r_1(\tau^*, n, \epsilon_{\text{TEP}}), \delta(\tau)/n)$, we get

$$\mathbb{E}_{\text{LDPC}\,[\lambda(x),\rho(x),n]} \left[ P_{\text{W}}^{\text{TEP}}(\mathtt{C}, \epsilon) \right]$$

$$\approx 1 - \mathcal{Q}\left( \frac{\left. \frac{\partial r_1(\tau,n,\epsilon)}{\partial\epsilon} \right|_{\substack{\tau=\tau^* \\ \epsilon=\epsilon_{\text{TEP}}}} \Delta\epsilon + \gamma_{\text{TEP}} n^{-1}}{\sqrt{\delta(\tau^*)/n}} \right) = \mathcal{Q}\left( \frac{\sqrt{n}(\epsilon_{\text{TEP}} - \epsilon)}{\alpha_{\text{TEP}}} + \frac{\gamma_{\text{TEP}}}{\sqrt{n}\ \delta(\tau^*)} \right), \tag{8}$$

where

$$\alpha_{\text{TEP}} = \sqrt{\delta(\tau^*)} \left( \frac{\partial r_1(\tau, \mathtt{n}, \epsilon)}{\partial \epsilon} \right)^{-1} \Bigg|_{\substack{\tau=\tau^* \\ \epsilon=\epsilon_{\text{TEP}}}}. \tag{9}$$

Finally, note that, since for $\mathtt{n} \to \infty$ we know that the TEP and the BP converge to the same solution, we can approximate $\alpha_{\text{TEP}} \approx \alpha_{\text{BP}}$. Besides, we have empirically observed that the variance of trajectories under BP and TEP decoding are quite similar so, for simplicity, we set $\delta(\tau^*)$ in (8) equal to $\delta(\tau^*)_{\text{BP}}$, whose analytic solution can be found in [16, 1]. Hence, we consider the TEP decoder expected evolution to estimate the parameter $\gamma_{\text{TEP}}$ in (8). In Figure 7(b), we compare the TEP performance for the regular $(3, 6)$ ensemble (solid lines) with the approximation in (8) (dashed lines), using the approximation $\alpha_{\text{TEP}} \approx \alpha_{\text{BP}} = 0.56036$, $\delta(\tau^*) \approx 0.0526$ and $\gamma_{\text{TEP}} \approx 0.3198$. We have plot the results for code lengths of $n = 2^9$ ($\circ$), $n = 2^{10}$ ($\square$), $n = 2^{11}$ ($\times$) and $2^{12}$ ($\triangleright$). As we can see, for the shortest code length, the model seems to slightly over-estimate the error probability, but this mismatch vanishes for the rest of the cases, obtaining a tight estimate.

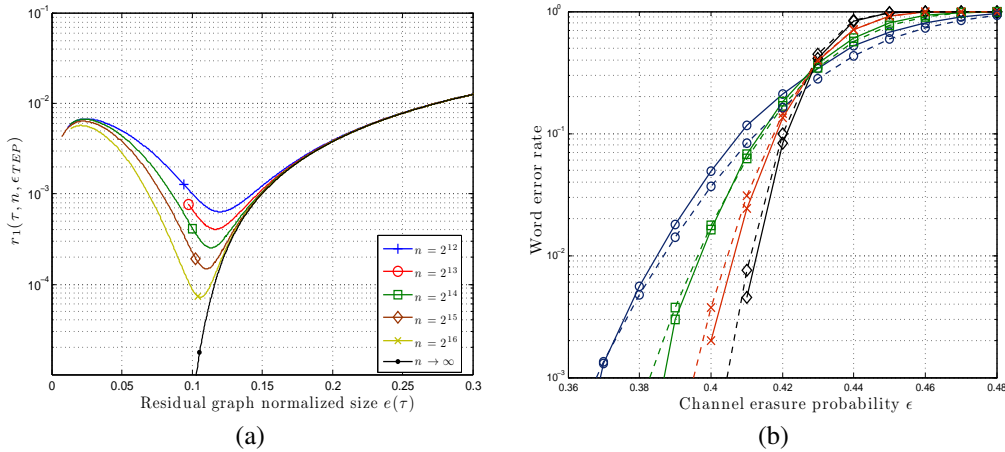

(a)             (b)

Figure 7: In (a), we plot the solution for $r_1(\tau)$ with respect to $e(t)$ for a $(3, 6)$ regular code at $\epsilon = \epsilon_{\text{BP}} = \epsilon_{\text{TEP}}$. In (b), we compare the TEP performance for the regular $(3, 6)$ ensemble (solid lines) with the approximation in (8) (dashed lines), using the approximation $\alpha_{\text{TEP}} \approx \alpha_{\text{BP}} = 0.56036$, $\delta(\tau^*) \approx 0.0526$ and $\gamma_{\text{TEP}} \approx 0.3198$. We have plot the results for code lengths of $\mathtt{n} = 2^9$ ($\circ$), $\mathtt{n} = 2^{10}$ ($\square$), $\mathtt{n} = 2^{11}$ ($\times$) and $\mathtt{n} = 2^{12}$ ($\triangleright$).

## 4 Conclusions

In this paper, we consider a tree structure for approximate inference in sparse graphical models using the EP algorithm. We have shown that, for finite-length LDPC sparse graphs, the accuracy of the marginal estimation with the method proposed significantly outperforms the BP estimate for the same graph. As a consequence, the decoding error rates are clearly improved. This result is remarkable in itself, as BP was considered the gold standard for LDPC decoding, and it was assumed that the long-range cycles and sparse nature of these factor graphs did not lend themselves for the application of more accurate approximate inference algorithms designed for dense graphs with short-range cycles. Additionally, the application of LDPC decoding showed us a different way of learning the tree structure that might be amenable for general factors.

## 5 Acknowledgments

This work was partially funded by Spanish government (Ministerio de Educación y Ciencia, TEC2009-14504-C02-01,02, Consolider-Ingenio 2010 CSD2008-00010), Universidad Carlos III (CCG10-UC3M/TIC-5304) and European Union (FEDER).

## Footnotes

[1]The BEC allows binary transmission, in which the bits are either erased with probability $\epsilon$ or arrive without error otherwise. The capacity for this channel is $1 - \epsilon$ and is achieved with equiprobable inputs [2].

# References

[1] Abdelaziz Amraoui, Andrea Montanari, Tom Richardson, and Rüdiger Urbanke. Finite-length scaling for iteratively decoded LDPC ensembles. *IEEE Transactions on Information Theory.*, 55(2):473–498, 2009.

[2] Thomas M. Cover and Joy A. Thomas. *Elements of Information Theory*. Wilson and Sons, New York, USA, 1991.

[3] Michael Luby, Michael Mitzenmacher, Amin Shokrollahi, Daniel Spielman, and Volker Stemann. Practical loss-resilient codes. In *Proceedings of the 29th annual ACM Symposium on Theory of Computing*, pages 150–159, 1997.

[4] Michael Luby, Michael Mitzenmacher, Amin Shokrollahi, Daniel Spielman, and Volker Stemann. Efficient erasure correcting codes. *IEEE Transactions on Information Theory*, 47(2):569–584, Feb. 2001.

[5] David J. C. MacKay. Good error-correcting codes based on very sparse matrices. *IEEE Transactions on Information Theory*, 45(2):399–431, 1999.

[6] David J. C. MacKay. *Information Theory, Inference, and Learning Algorithms*. Cambridge University Press, 2003.

[7] David J. C. MacKay and Radford M. Neal. Near Shannon limit performance of low density parity check codes. *Electronics Letters*, 32:1645–1646, 1996.

[8] T. Minka. Power EP. Technical report, MSR-TR-2004-149, 2004. http://research.microsoft.com/˜ minka/papers/.

[9] Thomas Minka and Yuan Qi. Tree-structured approximations by expectation propagation. In *Proceedings of the Neural Information Processing Systems Conference, (NIPS)*, 2003.

[10] Thomas P. Minka. Expectation Propagation for approximate Bayesian inference. In *Proceedings of the 17th Conference in Uncertainty in Artificial Intelligence (UAI 2001)*, pages 362–369. Morgan Kaufmann Publishers Inc., 2001.

[11] Pablo M. Olmos, Juan José Murillo-Fuentes, and Fernando Pérez-Cruz. Tree-structure expectation propagation for decoding LDPC codes over binary erasure channels. In *2010 IEEE International Symposium on Information Theory, ISIT, Austin, Texas*, 2010.

[12] P.M. Olmos, J.J. Murillo-Fuentes, and F. Pérez-Cruz. Tree-structure expectation propagation for LDPC decoding in erasure channels. *Submitted to IEEE Transactions on Information Theory*, 2011.

[13] P.M. Olmos, J.J. Murillo-Fuentes, and F. Pérez-Cruz. Tree-structured expectation propagation for decoding finite-length ldpc codes. *IEEE Communications Letters*, 15(2):235 –237, Feb. 2011.

[14] P. Oswald and A. Shokrollahi. Capacity-achieving sequences for the erasure channel. *IEEE Transactions on Information Theory*, 48(12):3017 – 3028, Dec. 2002.

[15] Tom Richardson and Ruediger Urbanke. *Modern Coding Theory*. Cambridge University Press, Mar. 2008.

[16] N. Takayuki, K. Kasai, and S. Kohichi. Analytical solution of covariance evolution for irregular LDPC codes. *e-prints*, November 2010.

[17] M. J. Wainwright, T. S. Jaakkola, and A. S. Willsk. Map estimation via agreement on (hyper)trees: Message-passing and linear-programming approaches. *IEEE Transactions on Information Theory*, 51(11):3697–3717, November 2005.

[18] Martin J. Wainwright and Michael I. Jordan. *Graphical Models, Exponential Families, and Variational Inference*. Foundations and Trends in Machine Learning, 2008.

[19] W. Weigerinck and T. Heskes. Fractional belief propagation. In S. Becker, S. Thrun, and K. Obermayer, editors, *Advances in Neural Information Processing Systems 15*, Cambridge, MA, December 2002. MIT Press.

[20] M. Welling, T. Minka, and Y.W. Teh. Structured region graphs: Morphing EP into GBP. In *UAI*, 2005.

[21] Nicholas C. Wormald. Differential equations for random processes and random graphs. *Annals of Applied Probability*, 5(4):1217–1235, 1995.

[22] J. S. Yedidia, W. T. Freeman, and Y. Weis. Constructing free-energy approximations and generalized belief propagation algorithms. *IEEE Transactions on Information Theory*, 51(7):2282–2312, July 2005.

